# Neural characterization in partially observed populations of spiking neurons

**Jonathan W. Pillow**          **Peter Latham**
Gatsby Computational Neuroscience Unit, UCL
17 Queen Square, London WC1N 3AR, UK
`pillow@gatsby.ucl.ac.uk`
`pel@gatsby.ucl.ac.uk`

## Abstract

Point process encoding models provide powerful statistical methods for understanding the responses of neurons to sensory stimuli. Although these models have been successfully applied to neurons in the early sensory pathway, they have fared less well capturing the response properties of neurons in deeper brain areas, owing in part to the fact that they do not take into account multiple stages of processing. Here we introduce a new twist on the point-process modeling approach: we include unobserved as well as observed spiking neurons in a joint encoding model. The resulting model exhibits richer dynamics and more highly nonlinear response properties, making it more powerful and more flexible for fitting neural data. More importantly, it allows us to estimate connectivity patterns among neurons (both observed and unobserved), and may provide insight into how networks process sensory input. We formulate the estimation procedure using variational EM and the wake-sleep algorithm, and illustrate the model's performance using a simulated example network consisting of two coupled neurons.

## 1 Introduction

A central goal of computational neuroscience is to understand how the brain transforms sensory input into spike trains, and considerable effort has focused on the development of statistical models that can describe this transformation. One of the most successful of these is the linear-nonlinear-Poisson (LNP) cascade model, which describes a cell's response in terms of a linear filter (or receptive field), an output nonlinearity, and an instantaneous spiking point process [1–5]. Recent efforts have generalized this model to incorporate spike-history and multi-neuronal dependencies, which greatly enhances the model's flexibility, allowing it to capture non-Poisson spiking statistics and joint responses of an entire population of neurons [6–10].

Point process models accurately describe the spiking responses of neurons in the early visual pathway to light, and of cortical neurons to injected currents. However, they perform poorly both in higher visual areas and in auditory cortex, and often do not generalize well to stimuli whose statistics differ from those used for fitting. Such failings are in some ways not surprising: the cascade model's stimulus sensitivity is described with a single linear filter, whereas responses in the brain reflect multiple stages of nonlinear processing, adaptation on multiple timescales, and recurrent feedback from higher-level areas. However, given its mathematical tractability and its accuracy in capturing the input-output properties of single neurons, the model provides a useful building block for constructing richer and more complex models of neural population responses.

Here we extend the point-process modeling framework to incorporate a set of unobserved or "hidden" neurons, whose spike trains are unknown and treated as hidden or latent variables. The unobserved neurons respond to the stimulus and to synaptic inputs from other neurons, and their spiking

activity can in turn affect the responses of the observed neurons. Consequently, their functional properties and connectivity can be inferred from data [11–18]. However, the idea is not to simply build a more powerful statistical model, but to develop a model that can help us learn something about the underlying structure of networks deep in the brain.

Although this expanded model offers considerably greater flexibility in describing an observed set of neural responses, it is more difficult to fit to data. Computing the likelihood of an observed set of spike trains requires integrating out the probability distribution over hidden activity, and we need sophisticated algorithms to find the maximum likelihood estimate of model parameters. Here we introduce a pair of estimation procedures based on variational EM (expectation maximization) and the wake-sleep algorithm. Both algorithms make use of a novel proposal density to capture the dependence of hidden spikes on the observed spike trains, which allows for fast sampling of hidden neurons' activity. In the remainder of this paper we derive the basic formalism and demonstrate its utility on a toy problem consisting of two neurons, one of which is observed and one which is designated "hidden". We show that a single-cell model used to characterize the observed neuron performs poorly, while a coupled two-cell model estimated using the wake-sleep algorithm performs much more accurately.

## 2 Multi-neuronal point-process encoding model

We begin with a description of the encoding model, which generalizes the LNP model to incorporate non-Poisson spiking and coupling between neurons. We refer to this as a generalized linear point-process (glpp) model[1] [8, 9]. For simplicity, we formulate the model for a pair of neurons, although it can be tractably applied to data from a moderate-sized populations ($\sim$10-100 neurons). In this section we do not distinguish between observed and unobserved spikes, but will do so in the next.

Let $\mathbf{x}_t$ denote the stimulus at time $t$, and $y_t$ and $z_t$ denote the number of spikes elicited by two neurons at $t$, where $t \in [0, T]$ is an index over time. Note that $\mathbf{x}_t$ is a vector containing all elements of the stimulus that are causally related to the (scalar) responses $y_t$ and $z_t$ at time $t$. Furthermore, let us assume $t$ takes on a discrete set of values, with bin size $\Delta$, i.e., $t \in \{0, \Delta, 2\Delta, \dots, T\}$. Typically $\Delta$ is sufficiently small that we observe only zero or one spike in every bin: $y_t, z_t \in \{0, 1\}$.

The conditional intensity (or instantaneous spike rate) of each cell depends on both the stimulus and the recent spiking history via a bank of linear filters. Let $\mathbf{y}_{[t-\tau,t)}$ and $\mathbf{z}_{[t-\tau,t)}$ denote the (vector) spike train histories at time $t$. Here $[t - \tau, t)$ refers to times between $t - \tau$ and $t - \Delta$, so $\mathbf{y}_{[t-\tau,t)} \equiv (y_{t-\tau}, y_{t-\tau+\Delta}, \dots, y_{t-2\Delta}, y_{t-\Delta})$ and similarly for $\mathbf{z}_{[t-\tau,t)}$. The conditional intensities for the two cells are then given by

$$\begin{aligned} \lambda_{yt} &= f(\mathbf{k}_y \cdot \mathbf{x}_t + \mathbf{h}_{yy} \cdot \mathbf{y}_{[t-\tau,t)} + \mathbf{h}_{yz} \cdot \mathbf{z}_{[t-\tau,t)}) \\ \lambda_{zt} &= f(\mathbf{k}_z \cdot \mathbf{x}_t + \mathbf{h}_{zz} \cdot \mathbf{z}_{[t-\tau,t)} + \mathbf{h}_{zy} \cdot \mathbf{y}_{[t-\tau,t)}) \end{aligned} \quad (1)$$

where $\mathbf{k}_y$ and $\mathbf{k}_z$ are linear filters representing each cell's receptive field, $\mathbf{h}_{yy}$ and $\mathbf{h}_{zz}$ are filters operating on each cell's own spike-train history (capturing effects like refractoriness and bursting), and $\mathbf{h}_{zy}$ and $\mathbf{h}_{yz}$ are a filters coupling the spike train history of each neuron to the other (allowing the model to capture statistical correlations and functional interactions between neurons). The "$\cdot$" notation represents the standard dot product (performing a summation over either index or time):

$$\begin{aligned} \mathbf{k} \cdot \mathbf{x}_t &\equiv \sum_i k_i x_{it} \\ \mathbf{h} \cdot \mathbf{y}_{[t-\tau,t)} &\equiv \sum_{t'=t-\tau}^{t-\Delta} h_{t'} y_{t'}, \end{aligned}$$

where the index $i$ run over the components of the stimuli (which typically are time points extending into the past). The second expression generalizes to $\mathbf{h} \cdot \mathbf{z}_{[t-\tau,t)}$.

The nonlinear function, $f$, maps the input to the instantaneous spike rate of each cell. We assume here that $f$ is exponential, although any monotonic convex function that grows no faster than expo-

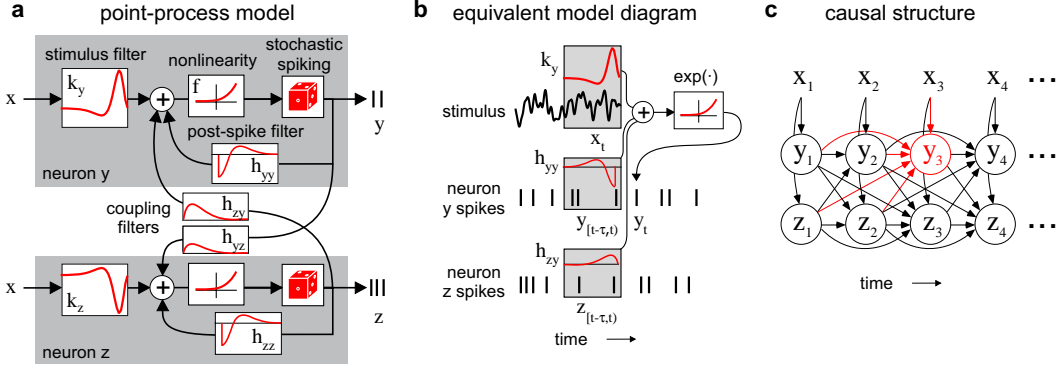

**Figure 1:** Schematic of generalized linear point-process (glpp) encoding model. **a**, Diagram of model parameters for a pair of coupled neurons. For each cell, the parameters consist of a stimulus filter (e.g., $\mathbf{k}_y$), a spike-train history filter ($\mathbf{h}_{yy}$), and a filter capturing coupling from the spike train history of the other cell ($\mathbf{h}_{zy}$). The filter outputs are summed, pass through an exponential nonlinearity, and drive spiking via an instantaneous point process. **b**, Equivalent diagram showing just the parameters of the neuron $y$, as used for drawing a sample $y_t$. Gray boxes highlight the stimulus vector $\mathbf{x}_t$ and spike train history vectors that form the input to the model on this time step. **c**, Simplified graphical model of the glpp causal structure, which allows us to visualize how the likelihood factorizes. Arrows between variables indicate conditional dependence. For visual clarity, temporal dependence is depicted as extending only two time bins, though in real data extends over many more. Red arrows highlight the dependency structure for a single time bin of the response $y_3$.

nentially is suitable [9]. Equation 1 is equivalent to $f$ applied to a linear convolution of the stimulus and spike trains with their respective filters; a schematic is shown in figure 1.

The probability of observing $y_t$ spikes in a bin of size $\Delta$ is given by a Poisson distribution with rate parameter $\lambda_{yt}\Delta$,

$$P(y_t|\lambda_{yt}) = \frac{(\lambda_{yt}\Delta)^{y_t}}{y_t!}e^{-\lambda_{yt}\Delta}, \tag{2}$$

and likewise for $P(z_t|\lambda_{zt})$. The likelihood of the full set of spike times is the product of conditionally independent terms,

$$P(Y, Z|X, \theta) = \prod_t P(y_t|\lambda_{yt})P(z_t|\lambda_{zt}), \tag{3}$$

where $Y$ and $Z$ represent the full spike trains, $X$ denotes the full set of stimuli, and $\theta \equiv \{\mathbf{k}_y, \mathbf{k}_z, \mathbf{h}_{yy}, \mathbf{h}_{zy}, \mathbf{h}_{zz}, \mathbf{h}_{yz}\}$ denotes the model parameters. This factorization is possible because $\lambda_{yt}$ and $\lambda_{zt}$ depend only on the process history up to time $t$, making $y_t$ and $z_t$ conditionally independent given the stimulus and spike histories up to $t$ (see Fig. 1c). If the response at time $t$ were to depend on both the past and future response, we would have a causal loop , preventing factorization and making both sampling and likelihood evaluation very difficult.

The model parameters can be tractably fit to spike-train data using maximum likelihood. Although the parameter space may be high-dimensional (incorporating spike-history dependence over many time bins and stimulus dependence over a large region of time and space), the negative log-likelihood is convex with respect to the model parameters, making fast convex optimization methods feasible for finding the global maximum [9]. We can write the log-likelihood simply as

$$\log P(Y, Z|X, \theta) = \sum_t (y_t \log \lambda_{yt} + z_t \log \lambda_{zt} - \Delta\lambda_{yt} - \Delta\lambda_{zt}) + c, \tag{4}$$

where $c$ is a constant that does not depend on $\theta$.

## 3 Generalized Expectation-Maximization and Wake-Sleep

Maximizing $\log P(Y, Z|X, \theta)$ is straightforward if both $Y$ and $Z$ are observed, but here we are interested in the case where $Y$ is observed and $Z$ is "hidden". Consequently, we have to average over $Z$. The log-likelihood of the observed data is given by

$$\mathcal{L}(\theta) \equiv \log P(Y|\theta) = \log \sum_Z P(Y, Z|\theta), \tag{5}$$

where we have dropped $X$ to simplify notation (all probabilities can henceforth be taken to also depend on $X$). This sum over $Z$ is intractable in many settings, motivating the use of approximate methods for maximizing likelihood. Variational expectation-maximization (EM) [20, 21] and the wake-sleep algorithm [22] are iterative algorithms for solving this problem by introducing a tractable approximation to the conditional probability over hidden variables,

$$Q(Z|Y, \phi) \approx P(Z|Y, \theta), \tag{6}$$

where $\phi$ denotes the parameter vector determining $Q$.

The idea behind variational EM can be described as follows. Concavity of the $\log$ implies a lower bound on the log-likelihood:

$$\begin{aligned} \mathcal{L}(\theta) &\geq \sum_Z Q(Z|Y, \phi) \log \frac{P(Y, Z|\theta)}{Q(Z|Y, \phi)} \\ &= \log P(Y|\theta) - D_{KL}\big(Q(Z|Y, \phi), P(Z|Y, \theta)\big), \end{aligned} \tag{7}$$

where $Q$ is any probability distribution over $Z$ and $D_{KL}$ is the Kullback-Leibler (KL) divergence between $Q$ and $P$ (using $P$ as shorthand for $P(Z|Y, \theta)$), which is always $\geq 0$. In standard EM, $Q$ takes the same functional form as $P$, so that by setting $\phi = \theta$ (the E-step), $D_{KL}$ is 0 and the bound is tight, since the right-hand-side of eq. 7 equals $\mathcal{L}(\theta)$. Fixing $\phi$, we then maximize the r.h.s. for $\theta$ (the M-step), which is equivalent to maximizing the expected complete-data log-likelihood (expectation taken w.r.t. $Q$), given by

$$E_{Q(Z|Y, \phi)}\big[\log P(Y, Z|\theta)\big] \equiv \sum_Z Q(Z|Y, \phi) \log P(Y, Z|\theta). \tag{8}$$

Each step increases a lower bound on the log-likelihood, which can always be made tight, so the algorithm converges to a fixed point that is a maximum of $\mathcal{L}(\theta)$. The variational formulation differs in allowing $Q$ to take a different functional form than $P$ (i.e., one for which eq. 8 is easier to maximize). The variational E-step involves minimizing $D_{KL}(Q, P)$ with respect to $\phi$, which remains positive if $Q$ does not approximate $P$ exactly; the variational M-step is unchanged from the standard algorithm.

In certain cases, it is easier to minimize the KL divergence $D_{KL}(P, Q)$ than $D_{KL}(Q, P)$, and doing so in place of the variational E-step above results in the wake-sleep algorithm [22]. In this algorithm, we fit $\phi$ by minimizing $D_{KL}(P, Q)$ averaged over $Y$, which is equivalent to maximizing the expectation

$$E_{P(Y, Z|\theta)}\big[\log Q(Z|Y, \phi)\big] \equiv \sum_{Y, Z} P(Y, Z|\theta) \log Q(Z|Y, \phi), \tag{9}$$

which bears an obvious symmetry to eq. 8. Thus, both steps of the wake-sleep algorithm involve maximizing an expected log-probability. In the "wake" step (identical to the M-step), we fit the true model parameters $\theta$ by maximizing (an approximation to) the log-probability of the observed data $Y$. In the "sleep" step, we fit $\phi$ by trying to find a distribution $Q$ that best approximates the conditional dependence of $Z$ on $Y$, averaged over the joint distribution $P(Y, Z|\theta)$. We can therefore think of the wake phase as learning a model of the data (parametrized by $\theta$), and the sleep phase as learning a consistent internal description of that model (parametrized by $\phi$).

Both variational-EM and the wake-sleep algorithm work well when $Q$ closely approximates $P$, but may fail to converge to a maximum of the likelihood if there is a significant mismatch. Therefore, the efficiency of these methods depends on choosing a good approximating distribution $Q(Z|Y, \phi)$ — one that closely matches $P(Z|Y, \theta)$. In the next section we show that considerations of the spike generation process can provide us with a good choice for $Q$.

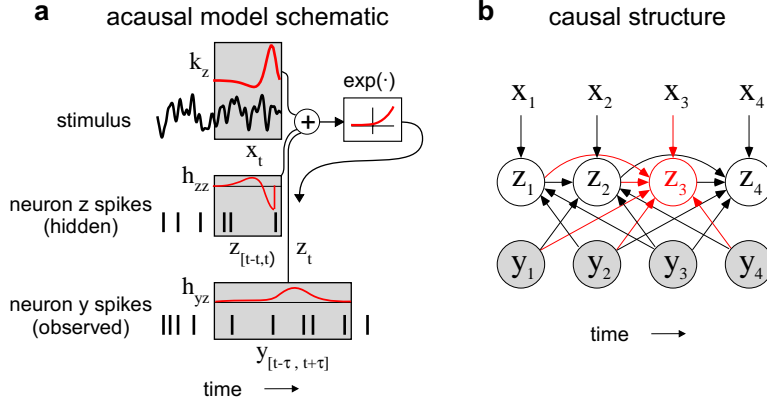

**a**     acausal model schematic       **b**     causal structure

**Figure 2:** Schematic diagram of the (acausal) model for the proposal density $Q(Z|Y,\phi)$, the conditional density on hidden spikes given the observed spike data. **a**, Conditional model schematic, which allows $z_t$ to depend on the observed response both before and after $t$. **b**, Graphical model showing causal structure of the acausal model, with arrows indicating dependency. The observed spike responses (gray circles) are no longer dependent variables, but regarded as fixed, external data, which is necessary for computing $Q(z_t|Y,\phi)$. Red arrows illustrate the dependency structure for a single bin of the hidden response, $z_3$.

## 4   Estimating the model with partially observed data

To understand intuitively why the true $P(Z|Y,\theta)$ is difficult to sample, and to motivate a reasonable choice for $Q(Z|Y,\phi)$, let us consider a simple example: suppose a single hidden neuron (whose full response is $Z$) makes a strong excitatory connection to an observed neuron (whose response is $Y$), so that if $z_t = 1$ (i.e., the hidden neuron spikes at time $t$), it is highly likely that $y_{t+1} = 1$ (i.e., the observed neuron spikes at time $t + 1$). Consequently, under the true $P(Z|Y,\theta)$, which is the probability over $Z$ in all time bins given $Y$ in all time bins, if $y_{t+1} = 1$ there is a high probability that $z_t = 1$. In other words, $z_t$ exhibits an acausal dependence on $y_{t+1}$. But this acausal dependence is not captured in Equation 3, which expresses the probability over $z_t$ as depending only on past events at time $t$, ignoring the future event $y_{t+1} = 1$.

Based on this observation — essentially, that the effect of future observed spikes on the probability of unobserved spikes depends on the connection strength between the two neurons — we approximate $P(Z|Y,\theta)$ using a separate point-process model $Q(Z|Y,\phi)$, which contains set of acausal linear filters from $Y$ to $Z$. Thus we have

$$\tilde{\lambda}_{zt} = \exp(\tilde{\mathbf{k}}_z \cdot \mathbf{x}_t + \tilde{\mathbf{h}}_{zz} \cdot \mathbf{z}_{[t-\tau,t)} \; + \; \tilde{\mathbf{h}}_{zy} \cdot \mathbf{y}_{[t-\tau,t+\tau)}). \qquad (10)$$

As above, $\tilde{\mathbf{k}}_z$, $\tilde{\mathbf{h}}_{zz}$ and $\tilde{\mathbf{h}}_{zy}$ are linear filters; the important difference is that $\tilde{\mathbf{h}}_{zy} \cdot \mathbf{y}_{[t-\tau,t+\tau)}$ is a sum over past and future time: from $t - \tau$ to $t + \tau - \Delta$. For this model, the parameters are $\phi = (\tilde{\mathbf{k}}_z, \tilde{\mathbf{h}}_{zz}, \tilde{\mathbf{h}}_{zy})$. Figure 2 illustrates the model architecture.

We now have a straightforward way to implement the wake-sleep algorithm, using samples from $Q$ to perform the wake phase (estimating $\theta$), and samples from $P(Y,Z|\theta)$ to perform the sleep phase (estimating $\phi$). The algorithm works as follows:

- **Wake:** Draw samples $\{Z_i\} \sim Q(Z|Y,\phi)$, where $Y$ are the observed spike trains and $\phi$ is the current set of parameters for the acausal point-process model $Q$. Evaluate the expected complete-data log-likelihood (eq. 8) using Monte Carlo integration:

$$E_Q\big[\log P(Y,Z|\theta)\big] = \lim_{N\to\infty} \frac{1}{N} \sum_{i=1}^{N} \log P(Y,Z_i|\theta). \qquad (11)$$

This is log-concave in $\theta$, meaning that we can efficiently find its global maximum to fit $\theta$.

- **Sleep:** Draw samples $\{Y_j, Z_j\} \sim P(Y, Z|\theta)$, the true encoding distribution with current parameters $\theta$. (Note these samples are pure "fantasy" data, drawn without reference to the observed $Y$). As above, compute the expected log-probability (eq. 9) using these samples:

$$E_{P(Y,Z|\theta)}\big[\log Q(Z|Y,\phi)\big] = \lim_{N\to\infty} \frac{1}{N}\sum_{i=1}^{N} \log Q(Z_j|Y_j,\phi), \qquad (12)$$

which is also log-concave and thus efficiently maximized for $\phi$.

One advantage of the wake-sleep algorithm is that each complete iteration can be performed using only a single set of samples drawn from $Q$ and $P$. A theoretical drawback to wake-sleep, however, is that the sleep step is not guaranteed to increase a lower-bound on the log-likelihood, as in variational-EM (wake-sleep minimizes the "wrong" KL divergence). We can implement variational-EM using the same approximating point-process model $Q$, but we now require multiple steps of sampling for a complete E-step. To perform a variational E-step, we draw samples (as above) from $Q$ and use them to evaluate both the KL divergence $D_{KL}\big(Q(Z|Y,\phi)||P(Z|Y,\theta)\big)$ and its gradient with respect to $\phi$. We can then perform noisy gradient descent to find a minimum, drawing a new set of samples for each evaluation of $D_{KL}(Q, P)$. The M-step is equivalent to the wake phase of wake-sleep, achievable with a single set of samples.

One additional use for the approximating point-process model $Q$ is as a "proposal" distribution for Metropolis-Hastings sampling of the true $P(Z|Y,\theta)$. Such samples can be used to evaluate the true log-likelihood, for comparison with the variational lower bound, and for noisy gradient ascent of the likelihood to examine how closely these approximate methods converge to the true ML estimate. For fully observed data, such samples also provide a useful means for measuring how much the entropy of one neuron's response is reduced by knowing the responses of its neighbors.

## 5 Simulations: a two-neuron example

To verify the method, we applied it to a pair of neurons (as depicted in fig. 1), simulated using a stimulus consisting of a long presentation of white noise. We denoted one of the neurons "observed" and the other "hidden". The parameters used for the simulation are depicted in fig. 3. The cells have similarly-shaped biphasic stimulus filters with opposite sign, like those commonly observed in ON and OFF retinal ganglion cells. We assume that the ON-like cell is observed, while the OFF-like cell is hidden. Both cells have spike-history filters that induce a refractory period following a spike, with a small peak during the relative refractory period that elicits burst-like responses. The hidden cell has a strong positive coupling filter $\mathbf{h}_{zy}$ onto the observed cell, which allows spiking activity in the hidden cell to excite the observed cell (despite the fact that the two cells receive opposite-sign stimulus input). For simplicity, we assume no coupling from the observed to the hidden cell [2]. Both types of filters were defined in a linear basis consisting of four raised cosines, meaning that each filter is specified by four parameters, and the full model contains 20 parameters (i.e., 2 stimulus filters and 3 spike-train filters).

Fig. 3b shows rasters of the two cells' responses to a repeated presentations of a 1s Gaussian white-noise stimulus with a framerate of 100Hz. Note that the temporal structure of the observed cell's response is strongly correlated with that of the hidden cell due to the strong coupling from hidden to observed (and the fact that the hidden cell receives slightly stronger stimulus drive).

Our first task is to examine whether a standard, single-cell glpp model can capture the mapping from stimuli to spike responses. Fig. 3c shows the parameters obtained from such a fit to the observed data, using 10s of the response to a non-repeating white noise stimulus (1000 samples, 251 spikes). Note that the estimated stimulus filter (red) has much lower amplitude than the stimulus filter of the true model (gray). Fig. 3d shows the parameters obtained for an observed and a hidden neuron, estimated using wake-sleep as described in section 4. Fig. 3e-f shows a comparison of the performance of the two models, indicating that the coupled model estimated with wake-sleep does a much better job of capturing the temporal structure of the observed neuron's response (accounting for 60% vs. 15% of

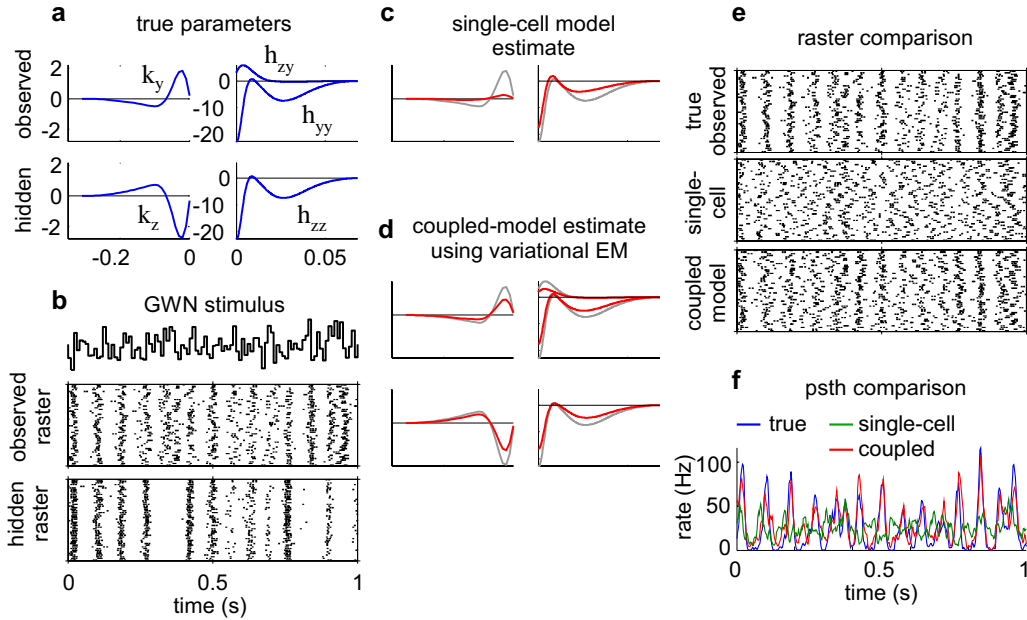

**Figure 3:** Simulation results. **a**, Parameters used for generating simulated responses. The top row shows the filters determining the input to the observed cell, while the bottom row shows those influencing the hidden cell. **b**, Raster of spike responses of observed and hidden cells to a repeated, 1s Gaussian white noise stimulus (top). **c**, Parameter estimates for a single-cell glpp model fit to the observed cell's response, using just the stimulus and observed data (estimates in red; true observed-cell filters in gray). **d**, Parameters obtained using wake-sleep to estimate a coupled glpp model, again using only the stimulus and observed spike times. **e**, Response raster of true observed cell (obtained by simulating the true two-cell model), estimated single-cell model and estimated coupled model. **f**, Peri-stimulus time histogram (PSTH) of the above rasters showing that the coupled model gives much higher accuracy predicting the true response.

the PSTH variance). The single-cell model, by contrast, exhibits much worse performance, which is unsurprising given that the standard glpp encoding model can capture only quasi-linear stimulus dependencies.

## 6   Discussion

Although most statistical models of spike trains posit a direct pathway from sensory stimuli to neuronal responses, neurons are in fact embedded in highly recurrent networks that exhibit dynamics on a broad range of time-scales. To take into account the fact that neural responses are driven by both stimuli and network activity, and to understand the role of network interactions, we proposed a model incorporating both hidden and observed spikes. We regard the observed spike responses as those recorded during a typical experiment, while the responses of unobserved neurons are modeled as latent variables (unrecorded, but exerting influence on the observed responses). The resulting model is tractable, as the latent variables can be integrated out using approximate sampling methods, and optimization using variational EM or wake-sleep provides an approximate maximum likelihood estimate of the model parameters. As shown by a simple example, certain settings of model parameters necessitate the incorporation unobserved spikes, as the standard single-stage encoding model does not accurately describe the data.

In future work, we plan to examine the quantitative performance of the variational-EM and wake-sleep algorithms, to explore their tractability in scaling to larger populations, and to apply them to real neural data. The model offers a promising tool for analyzing network structure and network-based computations carried out in higher sensory areas, particularly in the context where data are only available from a restricted set of neurons recorded within a larger population.

## Footnotes

[1]We adapt this terminology from "generalized linear model" (glm), a much more general class of models from the statistics literature [19]; this model is a glm whose distribution function is Poisson.

[2]Although the stimulus and spike-history filters bear a rough similarity to those observed in retinal ganglion cells, the coupling used here is unlike coupling filters observed (to our knowledge) between ON and OFF cells in retinal data; it is assumed purely for demonstration purposes.

# References

[1] I. Hunter and M. Korenberg. The identification of nonlinear biological systems: Wiener and hammerstein cascade models. *Biological Cybernetics*, 55:135–144, 1986.

[2] N. Brenner, W. Bialek, and R. de Ruyter van Steveninck. Adaptive rescaling optimizes information transmission. *Neuron*, 26:695–702, 2000.

[3] H. Plesser and W. Gerstner. Noise in integrate-and-fire neurons: From stochastic input to escape rates. *Neural Computation*, 12:367–384, 2000.

[4] E. J. Chichilnisky. A simple white noise analysis of neuronal light responses. *Network: Computation in Neural Systems*, 12:199–213, 2001.

[5] E. P. Simoncelli, L. Paninski, J. Pillow, and O. Schwartz. Characterization of neural responses with stochastic stimuli. In M. Gazzaniga, editor, *The Cognitive Neurosciences*, pages 327–338. MIT Press, 3rd edition, 2004.

[6] M. Berry and M. Meister. Refractoriness and neural precision. *Journal of Neuroscience*, 18:2200–2211, 1998.

[7] K. Harris, J. Csicsvari, H. Hirase, G. Dragoi, and G. Buzsaki. Organization of cell assemblies in the hippocampus. *Nature*, 424:552–556, 2003.

[8] W. Truccolo, U. T. Eden, M. R. Fellows, J. P. Donoghue, and E. N. Brown. A point process framework for relating neural spiking activity to spiking history, neural ensemble and extrinsic covariate effects. *J. Neurophysiol*, 93(2):1074–1089, 2004.

[9] L. Paninski. Maximum likelihood estimation of cascade point-process neural encoding models. *Network: Computation in Neural Systems*, 15:243–262, 2004.

[10] J. W. Pillow, J. Shlens, L. Paninski, A. Sher, A. M. Litke, and E. P. Chichilnisky, E. J. Simoncelli. Correlations and coding with multi-neuronal spike trains in primate retina. *SFN abstracts*, #768.9, 2007.

[11] D. Nykamp. Reconstructing stimulus-driven neural networks from spike times. *NIPS*, 15:309–316, 2003.

[12] D. Nykamp. Revealing pairwise coupling in linear-nonlinear networks. *SIAM Journal on Applied Mathematics*, 65:2005–2032, 2005.

[13] M. Okatan, M. Wilson, and E. Brown. Analyzing functional connectivity using a network likelihood model of ensemble neural spiking activity. *Neural Computation*, 17:1927–1961, 2005.

[14] L. Srinivasan, U. Eden, A. Willsky, and E. Brown. A state-space analysis for reconstruction of goal-directed movements using neural signals. *Neural Computation*, 18:2465–2494, 2006.

[15] D. Nykamp. A mathematical framework for inferring connectivity in probabilistic neuronal networks. *Mathematical Biosciences*, 205:204–251, 2007.

[16] J. E. Kulkarni and L Paninski. Common-input models for multiple neural spike-train data. *Network: Computation in Neural Systems*, 18(4):375–407, 2007.

[17] B. Yu, A. Afshar, G. Santhanam, S. Ryu, K. Shenoy, and M. Sahani. Extracting dynamical structure embedded in neural activity. *NIPS*, 2006.

[18] S. Escola and L. Paninski. Hidden Markov models applied toward the inference of neural states and the improved estimation of linear receptive fields. *COSYNE07*, 2007.

[19] P. McCullagh and J. Nelder. *Generalized linear models*. Chapman and Hall, London, 1989.

[20] A. Dempster, N. Laird, and R. Rubin. Maximum likelihood from incomplete data via the EM algorithm. *J. Royal Statistical Society, B*, 39(1):1–38, 1977.

[21] R. Neal and G. Hinton. A view of the EM algorithm that justifies incremental, sparse, and other variants. In M. I. Jordan, editor, *Learning in Graphical Models*, pages 355–368. MIT Press, Cambridge, 1999.

[22] GE Hinton, P. Dayan, BJ Frey, and RM Neal. The" wake-sleep" algorithm for unsupervised neural networks. *Science*, 268(5214):1158–1161, 1995.

